# Exponential Family Predictive Representations of State

**David Wingate**
Computer Science and Engineering
University of Michigan
wingated@umich.edu

**Satinder Singh**
Computer Science and Engineering
University of Michigan
baveja@umich.edu

## Abstract

In order to represent state in controlled, partially observable, stochastic dynamical systems, some sort of sufficient statistic for history is necessary. Predictive representations of state (PSRs) capture state as statistics of the future. We introduce a new model of such systems called the "Exponential family PSR," which defines as state the time-varying parameters of an exponential family distribution which models $n$ sequential observations in the future. This choice of state representation explicitly connects PSRs to state-of-the-art probabilistic modeling, which allows us to take advantage of current efforts in high-dimensional density estimation, and in particular, graphical models and maximum entropy models. We present a parameter learning algorithm based on maximum likelihood, and we show how a variety of current approximate inference methods apply. We evaluate the quality of our model with reinforcement learning by directly evaluating the control performance of the model.

## 1 Introduction

One of the basic problems in modeling controlled, partially observable, stochastic dynamical systems is representing and tracking state. In a reinforcement learning context, the state of the system is important because it can be used to make predictions about the future, or to control the system optimally. Often, state is viewed as an unobservable, latent variable, but models with *predictive representations of state* [4] propose an alternative: PSRs represent state as *statistics about the future*.

The original PSR models used the probability of specific, detailed futures called *tests* as the statistics of interest. Recent work has introduced the more general notion of using parameters that model the distribution of length $n$ futures as the statistics of interest [8]. To clarify this, consider an agent interacting with the system. It observes a series of observations $o_1...o_t$, which we call a *history* $h_t$ (where subscripts denote time). Given any history, there is some distribution over the next $n$ observations: $p(O_{t+1}...O_{t+n}|h_t) \equiv p(F^n|h_t)$ (where $O_{t+i}$ is the random variable representing an observation $i$ steps in the future, and $F^n$ is a mnemonic for *future*). We emphasize that this distribution directly models observable quantities in the system.

Instead of capturing state with tests, the more general idea is to capture state by directly modeling the distribution $p(F^n|h_t)$. Our central assumption is that the parameters describing $p(F^n|h_t)$ are sufficient for history, and therefore constitute state (as the agent interacts with the system, $p(F^n|h_t)$ changes because $h_t$ changes; therefore the parameters and hence state change). As an example of this, the Predictive Linear-Gaussian (PLG) model [8] assumes that $p(F^n|h_t)$ is jointly Gaussian; state therefore becomes its mean and covariance. Nothing is lost by defining state in terms of observable quantities: Rudary et al [8] proved that the PLG is formally equivalent to the latent-variable approach in linear dynamical systems. In fact, because the parameters are grounded, statistically consistent parameter estimators are available for PLGs.

Thus, as part of capturing state in a dynamical system in our method, $p(F^n|h_t)$ must be estimated. This is a density estimation problem. In systems with rich observations (say, camera images), $p(F^n|h_t)$ may have high dimensionality. As in all high-dimensional density estimation problems, structure must be exploited. It is therefore natural to connect to the large body of recent research dealing with high-dimensional density estimation, and in particular, graphical models.

In this paper, we introduce the *Exponential Family PSR* (EFPSR) which assumes that $p(F^n|h_t)$ is a standard exponential family distribution. By selecting the sufficient statistics of the distribution carefully, we can impose graphical structure on $p(F^n|h_t)$, and therefore make explicit connections to graphical models, maximum entropy modeling, and Boltzmann machines. The EFPSR inherits both the advantages and disadvantages of graphical exponential family models: inference and parameter learning in the model is generally hard, but all existing research on exponential family distributions is applicable (in particular, work on approximate inference).

Selecting the form of $p(F^n|h_t)$ and estimating its parameters to capture state is only half of the problem. We must also model the dynamical component, which describes the way that the parameters vary over time (that is, how the parameters of $p(F^n|h_t)$ and $p(F^n|h_{t+1})$ are related). We describe a method called "extend-and-condition," which generalizes many state update mechanisms in PSRs.

Importantly, the EFPSR has no hidden variables, but can still capture state, which sets it apart from other graphical models of sequential data. It is not directly comparable to latent-variable models such as HMMs, CRFs [3], or Maximum-entropy Markov Models (MEMMs) [5], for example. In particular, EM-based procedures used in the latent-variable models for parameter learning are unnecessary, and indeed, impossible. This is a consequence of the fact that the model is fully observed: all statistics of interest are directly related to observable quantities.

We refer the reader to [11] for an extended version of this paper.

## 2 The Exponential Family PSR

We now present the Exponential Family PSR (EFPSR) model. The next sections discuss the specifics of the central parts of the model: the state representation, and how we maintain that state.

### 2.1 Standard Exponential Family Distributions

We first discuss exponential family distributions, which we use because of their close connections to maximum entropy modeling and graphical models. We refer the reader to Jaynes [2] for detailed justification, but briefly, he states that the maximum entropy distribution "agrees with everything that is known, but carefully avoids assuming anything that is not known," which "is the fundamental property which justifies its use for inference." The standard exponential family distribution is the form of the maximum entropy distribution under certain constraints.

For a random variable $X$, a standard exponential family distribution has the form $p(X = x; s) = \exp\{s^T\phi(x) - Z(s)\}$, where $s$ is the canonical (or natural) vector of parameters and $\phi(x)$ is a vector of features of variable $x$. The vector $\phi(x)$ also forms the sufficient statistics of the distribution. The term $Z(s)$ is known as the log-partition function, and is a normalizing constant which ensures that $p(X; s)$ defines a valid distribution: $Z(s) = \log \int \exp\{s^T\phi(x)\}dx$. By carefully selecting the features $\phi(x)$, graphical structure may be imposed on the distribution.

### 2.2 State Representation and Dynamics

**State**. The EFPSR defines state as the parameters of an exponential family distribution modeling $p(F^n|h_t)$. To emphasize that these parameters represent state, we will refer to them as $s_t$:

$$p(F^n = f^n|h_t; s_t) = \exp\left\{s_t^\top \phi(f^n) - \log Z(s_t)\right\},\tag{1}$$

with both $\{\,\phi(f^n),\, s_t\,\} \in \mathbb{R}^{l\times 1}$. We emphasize that $s_t$ changes with history, but $\phi(f^n)$ does not.

**Maintaining State**. In addition to selecting the form of $p(F^n|h_t)$, there is a dynamical component: given the parameters of $p(F^n|h_t)$, how can we incorporate a new observation to find the parameters of $p(F^n|h_t, o_{t+1})$? Our strategy is to *extend and condition*, as we now explain.

**Extend.** We assume that we have the parameters of $p(F^n|h_t)$, denoted $s_t$. We *extend* the distribution of $F^n|h_t$ to include $O_{t+n+1}$, which forms a new variable $F^{n+1}|h_t$, and we assume it has the distribution $p(F^n, O_{t+n+1}|h_t) = p(F^{n+1}|h_t)$. This is a temporary distribution with $(n+1)d$ random variables. In order to add the new variable $O_{t+n+1}$, we must add new features which describe $O_{t+n+1}$ and its relationship to $F^n$. We capture this with a new feature vector $\phi^+(f^{n+1}) \in \mathbb{R}^{k \times 1}$, and define the vector $s_t^+ \in \mathbb{R}^{k \times 1}$ to be the parameters associated with this feature vector. We thus have the following form for the extended distribution:

$$p(F^{n+1} = f^{n+1}|h_t; s_t^+) = \exp\left\{s_t^{+\top}\phi^+(f^{n+}) - \log Z(s_t^+)\right\}.$$

To define the dynamics, we define a function which maps the current state vector to the parameters of the extended distribution. We call this the *extension function*: $s_t^+ = \text{extend}(s_t; \theta)$, where $\theta$ is a vector of parameters controlling the extension function (and hence, the overall dynamics).

The extension function helps govern the kinds of dynamics that the model can capture. For example, in the PLG family of work, a linear extension allows the model to capture linear dynamics [8], while a non-linear extension allows the model to capture non-linear dynamics [11].

**Condition.** Once we have extended the distribution to model the $n+1$'st observation in the future, we then condition on the *actual* observation $o_{t+1}$, which results in the parameters of a distribution over observations from $t+1$ through $t+n+1$: $s_{t+1} = \text{condition}(s_t^+, o_{t+1})$, which are precisely the statistics representing $p(F^n|h_{t+1})$, which is our state at time $t+1$.

By extending and conditioning, we can maintain state for arbitrarily long periods. Furthermore, for many choices of features and extension function, the overall extend-and-condition operation does not involve any inference, mean that tracking state is computationally efficient.

There is only one restriction on the extension function: we must ensure that after extending and conditioning the distribution, the resulting distribution can be expressed as: $p(F^n = f^n|h_{t+1}; s_{t+1}) = \exp\{s_{t+1}^\top \phi(f^n) - \log Z(s_{t+1})\}$. This looks like exactly like Eq. 1, which is the point: the feature vector $\phi$ did not change between timesteps, which means the form of the distribution does not change. For example, if $p(F^n|h_t)$ is a Gaussian, then $p(F^n|h_{t+1})$ will also be a Gaussian.

### 2.3 Representational Capacity

The EFPSR model is quite general. It has been shown that a number of popular models can be unified under the umbrella of the general EFPSR: for example, every PSR can be represented as an EFPSR (implying that every POMDP, MDP, and $k$-th order Markov model can also be represented as an EFPSR); and every linear dynamical system (Kalman filter) and some nonlinear dynamical systems can also be represented by an EFPSR. These different models are obtained with different choices of the features $\phi$ and the extension function, and are possible because many popular distributions (such as multinomials and Gaussians) are exponential family distributions [11].

## 3 The Linear-Linear EFPSR

We now choose specific features and extension function to generate an example model designed to be analytically tractable. We select a linear extension function, and we carefully choose features so that conditioning is always a linear operation. We restrict the model to domains in which the observations are vectors of binary random variables. The result is named the Linear-Linear EFPSR.

**Features.** Recall that the features $\phi()$ and $\phi^+()$ do not depend on time. This is equivalent to saying that the form of the distribution does not vary over time. If the features impose graphical structure on the distribution, it is also equivalent to saying that the form of the graph does not change over time. Because of this, we will now discuss how we can use a graph whose form is independent of time to help define structure on our distributions.

We construct the feature vectors $\phi()$ and $\phi^+()$ as follows. Let each $O_t \in \{0,1\}^d$; therefore, each $F^n|h_t \in \{0,1\}^{nd}$. Let $(F^n)^i$ be the $i$'th random variable in $F^n|h_t$. We assume that we have an undirected graph $G$ which we will use to create the features in the vector $\phi()$, and that we have another graph $G^+$ which we will use to define the features in the vector $\phi^+()$. Define $G = (V, E)$ where $V = \{1, ..., nd\}$ are the nodes in the graph (one for each $F^n|h_t{}^i$), and $(i, j) \in E$ are the

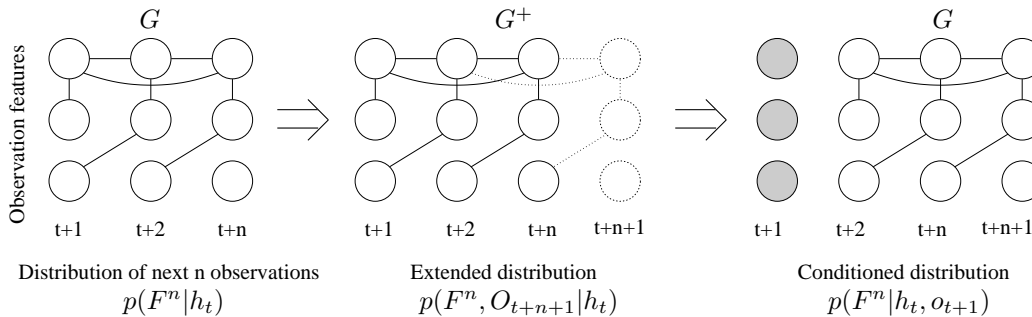

Figure 1: An illustration of extending and conditioning the distribution.

edges. Similarly, we define $G+ = (V+, E+)$ where $V+ = \{1, ..., (n+1)d\}$ are the nodes in the graph (one for each $(F^{n+1}|h_t)^i$), and $(i, j) \in E+$ are the edges. Neither graph depends on time.

To use the graph to define our distribution, we will let entries in $\phi$ be conjunctions of atomic observation variables (like the standard Ising model): for $i \in V$, there will be some feature $k$ in the vector such that $\phi(f_t)^k = f_t^i$. We also create one feature for each edge: if $(i, j) \in E$, then there will be some feature $k$ in the vector such that $\phi(f_t)^k = f_t^i f_t^j$. Similarly, we use $G^+$ to define $\phi^+()$.

As discussed previously, neither $G$ nor $G^+$ (equivalently, $\phi$ and $\phi^+$) can be arbitrary. We must ensure that after conditioning $G^+$, we recover $G$. To accomplish this, we ensure that both temporally shifted copies and conditioned versions of each feature exist in the graphs (seen pictorially in Fig. 1).

Because all features are either atomic variables or conjunctions of variables, conditioning the distribution can be done with an operation which is linear in the state (this is true even if the random variables are discrete or real-valued). We therefore define the linear conditioning operator $G(o_{t+1})$ to be a matrix which transforms $s_t^+$ into $s_{t+1}$: $s_{t+1} = G(o_{t+1})s_t^+$. See [11] for details.

**Linear extension**. In general, the function extend can take any form. We choose a linear extension:

$$s_t^+ = As_t + B$$

where $A \in \mathbb{R}^{k \times l}$ and $B \in \mathbb{R}^{k \times 1}$ are our model parameters. The combination of a linear extension and a linear conditioning operator can be rolled together into a single operation. Without loss of generality, we can permute the indices in our state vector such that $s_{t+1} = G(o_{t+1})(As_t + B)$. Note that although this is linear in the state, it is nonlinear in the observation.

## 4 Model Learning

We have defined our concept of state, as well as our method for tracking that state. We now address the question of learning the model from data. There are two things which can be learned in our model: the structure of the graph, and the parameters governing the state update. We briefly address each in the next two subsections. We assume we are given a sequence of $T$ observations, $[o_1 \cdots o_T]$, which we stack to create a sequence of samples from the $F^n|h_t$'s: $f_t|h_t = [o_{t+1} \cdots o_{t+n}|h_t]$.

### 4.1 Structure Learning

To learn the graph structure, we make the approximation of ignoring the dynamical component of the model. That is, we treat each $f_t$ as an observation, and try to estimate the density of the resulting unordered set, ignoring the $t$ subscripts (we appeal to density estimation because many good algorithms have been developed for structure induction). We therefore ignore temporal relationships *across* samples, but we preserve temporal relationships *within* samples. For example, if observation $a$ is always followed by observation $b$, this fact will be captured within the $f_t$'s.

The problem therefore becomes one of inducing graphical structure for a non-sequential data set, which is a problem that has already received considerable attention. In all of our experiments, we used the method of Della Pietra et. al [7]. Their method iteratively evaluates a set of candidate features and adds the one with highest expected gain in log-likelihood. To enforce the temporal

invariance property, whenever we add a feature, we also add all of the temporally shifted copies of that feature, as well as the conditioned versions of that feature.

## 4.2 Maximum Likelihood Parameter Estimation

With the structure of the graph in place, we are left to learn the parameters $A$ and $B$ of the state extension. It is now useful that our state is defined in terms of observable quantities, for two reasons: first, because everything in our model is observed, EM-style procedures for estimating the parameters of our model are not needed, simply because there are no unobserved variables over which to take expectations. Second, when trying to learn a sequence of states ($s_t$'s) given a long trajectory of futures ($f_t$'s), each $f_t$ is a sample of information directly from the distribution we're trying to model. Given a parameter estimate, an initial state $s_0$, and a sequence of observations, the sequence of $s_t$'s is completely determined. This will be a key element to our proposed maximum-likelihood learning algorithm.

Although the sequence of state vectors $s_t$ are the parameters defining the distributions $p(F^n|h_t)$, they are **not** the model parameters – that is, we cannot freely select them. Instead, the model parameters are the parameters $\theta$ which govern the extension function. This is a significant difference from standard maximum entropy models, and stems from the fact that our overall problem is that of modeling a dynamical system, rather than just density estimation.

The likelihood of the training data is $p(o_1, o_2...o_T) = \prod_{t=1}^{T} p(o_t|h_t)$. We will find it more convenient to measure the likelihood of the corresponding $f_t$'s: $p(o_1, o_2...o_T) \approx n \prod_{t=1}^{T} p(f_t|h_t)$ (the likelihoods are not the same because the likelihood of the $f_t$'s counts a single observation $n$ times; the approximate equality is because the first $n$ and last $n$ are counted fewer than $n$ times).

The expected log-likelihood of the training $f_t$'s under the model defined in Eq. 1 is

$$\mathcal{LL} = \frac{1}{T} \left( \sum_{t=1}^{T} -s_t^\top \phi(f_t) - \log Z(s_t) \right) \tag{2}$$

Our goal is to maximize this quantity. Any optimization method can be used to maximize the log-likelihood. Two popular choices are gradient ascent and quasi-Newton methods, such as (L-)BFGS. We use both, for different problems (as discussed later). However, both methods require the gradient of the likelihood with respect to the parameters, which we will now compute.

Using the chain rule of derivatives, we can compute the derivative with respect to the parameters $A$:

$$\frac{\partial \mathcal{LL}}{\partial A} = \sum_{t=1}^{T} \frac{\partial \mathcal{LL}}{\partial s_t}^\top \frac{\partial s_t}{\partial A} \tag{3}$$

First, we compute the derivative of the log-likelihood with respect to each state:

$$\frac{\partial \mathcal{LL}}{\partial s_t} = \frac{\partial}{\partial s_t} \left[ -s_t^\top \phi(f_t) - \log Z(s_t) \right] = \mathrm{E}_{s_t}[\phi(F^n|h_t)] - \phi(f_t) \equiv \delta_t \tag{4}$$

where $\mathrm{E}_{s_t}[\phi(F^n|h_t)] \in \mathbb{R}^{l \times 1}$ is the vector of expected sufficient statistics at time $t$. Computing this is a standard inference problem in exponential family models, as discussed in Section 5. This gradient tells us that we wish to adjust each state to make the expected features of the next $n$ observations closer to the observed features however, we cannot adjust $s_t$ directly; instead, we must adjust it implicitly by adjusting the transition parameters $A$ and $B$.

We now compute the gradients of the state with respect to each parameter:

$$\frac{\partial s_t}{\partial A} = \frac{\partial}{\partial A} G(o_{t+1}) (As_{t-1} + B) = G(o_{t+1}) \left( A \frac{\partial s_{t-1}}{\partial A} + s_{t-1}^\top \otimes I \right).$$

where $\otimes$ is the Kronecker product, and $I$ is an identity matrix the same size as $A$. The gradients of the state with respect to $B$ are given by

$$\frac{\partial s_t}{\partial B} = \frac{\partial}{\partial B} G(o_{t+1}) (As_{t-1} + B) = G(o_{t+1}) \left( A \frac{\partial s_{t-1}}{\partial B} + I \right)$$

These gradients are temporally recursive – they implicitly depend on gradients from all previous timesteps. It might seem prohibitive to compute them: must an algorithm examine all past $t_1 \cdots t_{t-1}$ data points to compute the gradient at time $t$? Fortunately, the answer is no: the necessary statistics can be computed in a recursive fashion as the algorithm walks through the data.

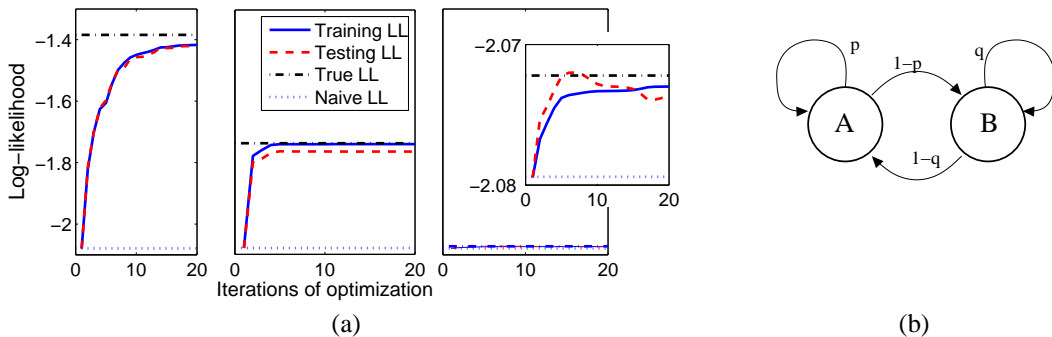

(a)                               (b)

Figure 2: Results on two-state POMDPs. The right shows the generic model used. By varying the transition and observation probabilities, three different POMDPs were generated. The left shows learning performance on the three models. Likelihoods for naive predictions are shown as a dotted line near the bottom; likelihoods for optimal predictions are shown as a dash-dot line near the top.

| Problem | # of states | # of obs. | # of actions | Naive LL | True LL | Training set LL | Training set % | Test set LL | Test set % |
|---------|-------------|-----------|--------------|----------|---------|-----------------|----------------|-------------|------------|
| Paint   | 16          | 2         | 4            | 6.24     | 4.66    | 4.67            | **99.7**       | 4.66        | **99.9**   |
| Network | 7           | 2         | 4            | 6.24     | 4.49    | 4.50            | **99.5**       | 4.52        | **98.0**   |
| Tiger   | 2           | 2         | 3            | 6.24     | 5.23    | 5.24            | **92.4**       | 5.25        | **86.0**   |

Figure 3: Results on standard POMDPs. See text for explanation.

## 5 Inference

In order to compute the gradients needed for model learning, the expected sufficient statistics $E[\phi(F^n|h_t)]$ at each timestep must be computed (see Eq. 4):

$$E\left[\phi(F^n|h_t)\right] = \int \phi(f_t)p(F^n|h_t)df_t = \nabla Z(s).$$

This quantity, also known as the *mean parameters*, is of central interest in standard exponential families, and has several interesting properties. For example, each possible set of canonical parameters $s$ induces one set of mean parameters; assuming that the features are linearly independent, each set of valid mean parameters is uniquely determined by one set of canonical parameters [9].

Computing these marginals is an inference problem. This is repeated $T$ times (the number of samples) in order to get one gradient, which is then used in an outer optimization loop; because inference must be repeatedly performed in our model, computational efficiency is a more stringent requirement than accuracy. In terms of inference, our model inherits all of the properties of graphical models, for better and for worse. Exact inference in our model is generally intractable, except in the case of fully factorized or tree-structured graphs. However, many approximate algorithms exist: there are variational methods such as naive mean-field, tree-reweighted belief propagation, and log-determinant relaxations [10]; other methods include Bethe-Kikuchi approximations, expectation propagation, (loopy) belief propagation, MCMC methods, and contrastive divergence [1].

## 6 Experiments and Results

Two sets of experiments were conducted to evaluate the quality of our model and learning algorithm. The first set tested whether the model could capture exact state, given the correct features and exact inference. We evaluated the learned model using exact inference to compute the exact likelihood of the data, and compared to the true likelihood. The second set tested larger models, for which exact inference is not possible. For the second set, bounds can be provided for the likelihoods, but may be so loose as to be uninformative. How can we assess the quality of the final model? One objective gauge is control performance: if the model has a reward signal, reinforcement learning can be used to determine an optimal policy. Evaluating the reward achieved becomes an objective measure of model quality, even though approximate likelihood is the learning signal.

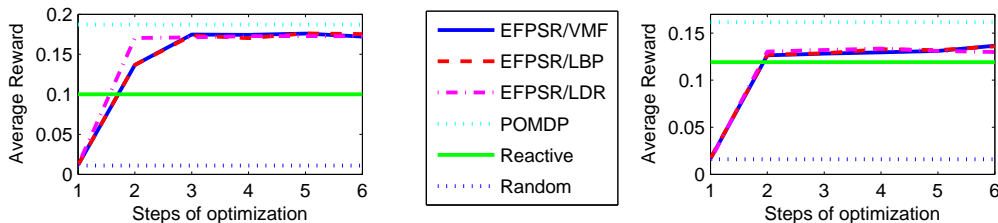

Figure 4: Results on Cheesemaze (left) and Maze 4x3 (right) for different inference methods.

**First set.** We tested on three two-state problems, as well as three small, standard POMDPs. For each problem, training and test sets were generated (using a uniformly random policy for controlled systems). We used 10,000 samples, set $n = 3$ and used structure learning as explained in Section 4.1. We used exact inference to compute the $\mathrm{E}[\phi(F^n|h_t)]$ term needed for the gradients. We optimized the likelihood using BFGS. For each dataset, we computed the log-likelihood of the data under the true model, as well as the log-likelihood of a "naive" model, which assigns uniform probability to every possible observation. We then learned the best model possible, and compared the final log-likelihood under the learned and true models.

Figure 2 (a) shows results for three two-state POMDPs with binary observations. The left panel of Fig. 2 (a) shows results for a two-state MDP. The likelihood of the learned model closely approaches the likelihood of the true model (although it does not quite reach it; this is because the model has trouble modeling deterministic observations, because the weights in the exponential need to be infinitely large [or small] to generate a probability of one [or zero]). The middle panel shows results for a moderately noisy POMDP; again, the learned model is almost perfect. The third panel shows results for a very noisy POMDP, in which the naive and true LLs are very close; this indicates that prediction is difficult, even with a perfect model.

Figure 3 shows results for three standard POMDPs, named Paint, Network and Tiger[1]. The table conveys similar information to the graphs: naive and true log-likelihoods, as well as the log-likelihood of the learned models (on both training and test sets). To help interpret the results, we also report a percentage (highlighted in bold), which indicates the amount of the likelihood gap (between the naive and true models) that was captured by the learned model. Higher is better; again we see that the learned models are quite accurate, and generalize well.

**Second set.** We also tested on a two more complicated POMDPs called Cheesemaze and Maze 4x3[1]. For both problems, exact inference is intractable, and so we used approximate inference. We experimented with loopy belief propagation (LBP) [12], naive mean field (or variational mean field, VMF), and log-determinant relaxations (LDR) [10]. Since the VMF and LDR bounds on the log-likelihood were so loose (and LBP provides no bound), it was impossible to assess our model by an appeal to likelihood. Instead, we opted to evaluate the models based on control performance.

We used the Natural Actor Critic (or NAC) algorithm [6] to test our model (see [11] for further experiments). The NAC algorithm requires two things: a stochastic, parameterized policy which operates as a function of state, and the gradients of the log probability of that policy. We used a softmax function of a linear projection of the state: the probability of taking action $a_i$ from state $s_t$ given the policy parameters $\theta$ is: $p(a_i; s_t, \theta) = \exp\left\{s_t^\top \theta_i\right\} / \sum_{j=1}^{|\mathcal{A}|} \exp\left\{s_t^\top \theta_j\right\}$. The parameters $\theta$ are to be determined. For comparison, we also ran the NAC planner with the POMDP belief state: we used the same stochastic policy and the same gradients, but we used the belief state of the true POMDP in place of the EFPSR's state ($s_t$). We also tested NAC with the first-order Markov assumption (or reactive policy) and a totally random policy.

**Results.** Figure 4 shows the results for Cheesemaze. The left panel shows the best control performance obtained (average reward per timestep) as a function of steps of optimization. The "POMDP" line shows the best reward obtained using the true belief state as computed under the true model, the "Random" line shows the reward obtained with a random policy, and the "Reactive" line shows the best reward obtained by using the observation as input to the NAC algorithm. The lines "VMF," "LBP," and "LDR" correspond to the different inference methods.

The EFPSR models all start out with performance equivalent to the random policy (average reward of 0.01), and quickly hop to of 0.176. This is close to the average reward of using the true POMDP state at 0.187. The EFPSR policy closes about 94% of the gap between a random policy and the policy obtained with the true model. Surprisingly, only a few iterations of optimization were necessary to generate a usable state representation. Similar results hold for the Maze 4x3 domain, although the improvement over the first order Markov model is not as strong: the EFPSR closes about 77.8% of the gap between a random policy and the optimal policy. We conclude that the EFPSR has learned a model which successfully incorporates information from history into the state representation, and that it is this information which the NAC algorithm uses to obtain better-than-reactive performance. This implies that the model and learning algorithm are useful even with approximate inference methods, and even in cases where we cannot compare to the exact likelihood.

## 7    Conclusions

We have presented the Exponential Family PSR, a new model of controlled, stochastic dynamical systems which provably unifies other models with predictively defined state. We have also discussed a specific member of the EFPSR family, the Linear-Linear EFPSR, and a maximum likelihood learning algorithm. We were able to learn almost perfect models of several small POMDP systems, both from a likelihood perspective and from a control perspective. The biggest drawback is computational: the repeated inference calls make the learning process very slow. Improving the learning algorithm is an important direction for future research. While slow, the learning algorithm generates models which can be accurate in terms of likelihood and useful in terms of control performance.

## Acknowledgments

David Wingate was supported under a National Science Foundation Graduate Research Fellowship. Satinder Singh was supported by NSF grant IIS-0413004. Any opinions, findings, and conclusions or recommendations expressed in this material are those of the authors and do not necessarily reflect the views of the NSF.

## Footnotes

[1]From Tony Cassandra's POMDP repository at http://www.cs.brown.edu/research/ai/pomdp/index.html

## References

[1]  G. E. Hinton. Training products of experts by minimizing contrastive divergence. *Neural Computation*, 14(8):1771–1800, 2002.

[2]  E. T. Jaynes. Notes on present status and future prospects. In W. Grandy and L. Schick, editors, *Maximum Entropy and Bayesian Methods*, pages 1–13, 1991.

[3]  J. Lafferty, A. McCallum, and F. Pereira. Conditional random fields: Probabilistic models for segmenting and labeling sequence data. In *International Conference on Machine Learning (ICML)*, 2001.

[4]  M. L. Littman, R. S. Sutton, and S. Singh. Predictive representations of state. In *Neural Information Processing Systems (NIPS)*, pages 1555–1561, 2002.

[5]  A. McCallum, D. Freitag, and F. Pereira. Maximum entropy Markov models for information extraction and segmentation. In *International Conference on Machine Learning (ICML)*, pages 591–598, 2000.

[6]  J. Peters, S. Vijayakumar, and S. Schaal. Natural actor-critic. In *European Conference on Machine Learning (ECML)*, pages 280–291, 2005.

[7]  S. D. Pietra, V. D. Pietra, and J. Lafferty. Inducing features of random fields. *IEEE Transactions on Pattern Analysis and Machine Intelligence*, 19(4):380–393, 1997.

[8]  M. Rudary, S. Singh, and D. Wingate. Predictive linear-Gaussian models of stochastic dynamical systems. In *Uncertainty in Artificial Intelligence (UAI)*, pages 501–508, 2005.

[9]  M. J. Wainwright and M. I. Jordan. Graphical models, exponential families, and variational inference. Technical Report 649, UC Berkeley, 2003.

[10] M. J. Wainwright and M. I. Jordan. Log-determinant relaxation for approximate inference in discrete Markov random fields. *IEEE Transactions on Signal Processing*, 54(6):2099–2109, 2006.

[11] D. Wingate. *Exponential Family Predictive Representations of State*. PhD thesis, University of Michigan, 2008.

[12] J. S. Yedida, W. T. Freeman, and Y. Weiss. Understanding belief propagation and its generalizations. Technical Report TR-2001-22, Mitsubishi Electric Research Laboratories, 2001.

